# Feedback control guides credit assignment in recurrent neural networks

**Klara Kaleb**
Department of Bioengineering
Imperial College London
London, UK
klara.kaleb18@imperial.ac.uk

**Barbara Feulner**
Department of Bioengineering
Imperial College London
London, UK

**Juan A. Gallego**
Department of Bioengineering
Imperial College London
London, UK

**Claudia Clopath**
Department of Bioengineering
Imperial College London
London, UK

## Abstract

How do brain circuits learn to generate behaviour? While significant strides have been made in understanding learning in artificial neural networks, applying this knowledge to biological networks remains challenging. For instance, while backpropagation is known to perform accurate credit assignment of error in artificial neural networks, how a similarly powerful process can be realized within the constraints of biological circuits remains largely unclear. One of the major challenges is that the brain's extensive recurrent connectivity requires the propagation of error through both space and time, a problem that is notoriously difficult to solve in vanilla recurrent neural networks. Moreover, the extensive feedback connections in the brain are known to influence forward network activity, but the interaction between feedback-driven activity changes and local, synaptic plasticity-based learning is not fully understood. Building on our previous work modelling motor learning, this work investigates the mechanistic properties of pre-trained networks with feedback control on a standard motor task. We show that feedback control of the ongoing recurrent network dynamics approximates the optimal first-order gradient with respect to the network activities, allowing for rapid, ongoing movement correction. Moreover, we show that trial-by-trial adaptation to a persistent perturbation using a local, biologically plausible learning rule that integrates recent activity and error feedback is both more accurate and more efficient *with* feedback control during learning, due to the decoupling of the recurrent network dynamics *and* the injection of an adaptive, second-order gradient into the network dynamics. Thus, our results suggest that feedback control may *guide* credit assignment in biological recurrent neural networks, enabling both rapid and efficient learning in the brain.

## 1 Introduction

Despite the "unreasonable effectiveness" of the backpropagation (BP) algorithm (Rumelhart et al., 1986), learning in the brain must leverage a different solution (Lillicrap & Santoro, 2019; Lillicrap et al., 2020). For instance, the brain's extensive recurrent connectivity (Felleman & Van Essen, 1991; Douglas & Martin, 2004) requires propagation of error through both space *and* time. Learning in vanilla recurrent neural networks (RNNs) with Backpropagation through time (BPTT) (Werbos, 1990; Mozer et al., 1995; Robinson & Fallside, 1987) is known to be difficult due to the vanishing

and exploding gradient problem (Pascanu et al., 2013; Hochreiter & Schmidhuber, 1997). This challenge is excarbated under biological constraints, which require learning rules with significant approximations of the full temporal memory (Marschall et al., 2020; Bellec et al., 2020; Murray, 2019; Liu et al., 2021), and often lead to an undesirable generalization gap (Liu et al., 2022).

In addition to intra-layer recurrence, the brain also features extensive feedback connections from top-down sources (Peters & Payne, 1993; Peters et al., 1994) that can modulate network activity in lower layers (Przybyszewski, 1998; Girard et al., 2001; De Pasquale & Sherman, 2011; Briggs & Usrey, 2011; Jordan & Keller, 2020; Keller et al., 2020) and improve bottom-up information processing (Gilbert & Li, 2013; Manita et al., 2015; Fyall et al., 2017). This multiplicity of simultaneously active pathways contrasts vanilla RNNs, where feedback connections are used exclusively for gradient propagation.

It is precisely this simultaneous integration of several streams of information that is thought to mediate ongoing movement correction in both humans and non-human primates (Thoroughman & Shadmehr, 2000; Perich et al., 2018; Krakauer et al., 2000). For example, when subjects receive distorted feedback during a learned reaching movement, such as an unexpected force acting on the limb, they are able to make movement corrections already at the first trial (Thoroughman & Shadmehr, 2000). Moreover, they rapidly adapt to the mismatch between the observed and expected movement trajectory upon further perturbed trials, requiring no more online corrections (Krakauer et al., 2000). Recent data-driven modelling work (Feulner et al., 2022) proposed an elegant solution to this problem by integrating feedback control into the recurrent network dynamics, and pre-training the whole network on a stereotypical motor task. The resulting networks showed rapid adaptation to task perturbation during ongoing movement and further improvement in the task performance upon persistent perturbation using a *feedback-driven, local learning rule*, and thus reproduce key behavioural and neurophysiological findings. However, the relationship between this local feedback-driven learning and gradient descent, or any of its approximations, remains unclear.

In this work, we investigate the mechanistic properties of such recurrent networks pre-trained with feedback control on a stereotypical motor task. Our key findings are listed as follows:

1. Feedback control allows for approximate learning in the activity space 3.2.

2. Feedback control enables increased accuracy of approximate, local learning rules in the recurrent layer due to the "decoupling" of the network from its past activity 3.4.

3. Feedback control enables more efficient weight updates during task adaptation due to the implicit incorporation of adaptive, second-order gradient into the network dynamics 3.5.

## Related work

In most modelling studies, the role of feedback connections is restricted to gradient signal propagation only (Bellec et al., 2020; Murray, 2019; Payeur et al., 2021; Liu et al., 2021). The need for separation seems clear: in the studies where this restriction is lifted, the plasticity mechanisms involved often require tight coordination (Scellier & Bengio, 2017; Whittington & Bogacz, 2017; Sacramento et al., 2018; Payeur et al., 2021; Podlaski & Machens, 2020) that may be hard to implement and maintain in the brain. Furthermore, the relative influence of feedback may still be kept small to avoid interference with the forward inference.

Nevertheless, several other computational studies have shown that a stronger influence of feedback on the network activities may actually aid network training (Gilra & Gerstner, 2017; Denève et al., 2017; Alemi et al., 2018; Bourdoukan & Deneve, 2015; Meulemans et al., 2021, 2022a,b, 2020). For example, Meulemans et al. (2020) formally characterize the link between previously proposed Target Propagation (Bengio, 2014; Lee et al., 2015) to more efficient learning with second-order optimization methods (Gauss, 1877). We also highlight Meulemans et al. (2022b), where the authors formally derive the relationship between gradient-based learning and optimal control through feedback activity influence for equilibrium systems. However, despite the success on a number of difficult problems, it remains unknown how well these findings extend to out-of-equilibrium dynamics commonly found in the brain. In contrast, Feulner et al. (2022) takes a more agnostic approach by pre-training recurrent networks with feedback control of their output out-of-equilibrium and under non-stationary conditions. However, while the resulting networks and the local learning algorithm reproduce key behavioural

and neurophysiological findings from both humans and non-human primates, their relationship with gradient descent remains unclear.

## 2 Methods

### 2.1 Training stages

In this work, we used a modelling setup built to reproduce motor adaption experiments in monkeys (Perich et al., 2018) and similar to that used in Feulner et al. (2022) (see Appendix A.6 for key differences). We construct a model that mimics both the movement repertoire and the flexibility of the motor cortex by applying a two-stage training strategy: pre-training followed by fine-tuning.

**Pre-training.** To obtain stereotypical motor movement dynamics, we first train the recurrent networks with feedback control on a synthetic *instructed delay centre-out reaching* task using BPTT (Werbos, 1990). Here, the task is to produce a broad set of two-dimensional reaching velocity trajectories of varying lengths, following an instructed delay phase (see Section A.1 for more details). A visualization of a sub-task with 8 equidistant targets can be seen in Figure 1a-b.

**Fine-tuning.** To probe the recurrent network's ability to adapt to task perturbations following initial pre-training, we replicate a classic visuomotor (VR) perturbation paradigm (Perich et al., 2018; Krakauer et al., 2000) in which the model output is rotated around the centre by a certain fixed angle. Motivated by progressive adaptations to such persistent perturbations in the experimental studies (Perich et al., 2018), in our work perturbation fine-tuning occurs through a local, biologically plausible learning rule.

### 2.2 Network architecture

We train the recurrent neural networks with a single recurrent hidden layer whose activity is described by the dynamics in Equation 1. Here, a hidden layer unit $h_j$ receives the stimulus signal $s^t$, the previous network activity post-nonlinearity $\Phi(h_i)$ and the positional error $\epsilon_i$ from the previous time step i.e. $(t-1)$.

$$\tau \dot{h_j} = \left( -h_j + \sum_i W_{ji}^{in} s_i + \sum_i W_{ji}^h \Phi(h_i) + \sum_k W_{ji}^{fb} \epsilon_i + b_j^h \right) \tag{1}$$

In Equation 1 $\Phi$ represents the ReLU function, and $\boldsymbol{W}^{fb}$ are the backward projecting feedback weights. The 2D output velocity $\boldsymbol{v}$ of the recurrent network is obtained through a linear readout of its hidden layer activity post-nonlinearity, where $a_j = \Phi(h_j)$:

$$v_k = W_{kj}^o a_j + b_k^o \tag{2}$$

### 2.3 Pre-training procedure

We pre-train $\boldsymbol{W}^{in}, \boldsymbol{W}^h, \boldsymbol{W}^{fb}, \boldsymbol{W}^o, \boldsymbol{b}^h$ and $\boldsymbol{b}^o$ using BPTT (Werbos, 1990) with mean squared error (MSE) over the integrated position space using Adam (Kingma & Ba, 2014). To encourage the networks to learn to use the feedback layer to correct their output online, we use randomly distributed velocity perturbations (see Section A.3 for more details). The 2D error feedback signal $\epsilon_k$ projected back to the network with feedback weights $\boldsymbol{W}^{fb}$ is the derivative of the MSE of the integrated position $p_k^t = p_k^{t-1} + dt \cdot v_k^t$, i.e. $\epsilon_k^t = p_k^{t*} - p_k^t$, where $p_k^{t*}$ is the task target.

### 2.4 Adaptation and feedback-driven plasticity rule

To assess how the feedback integration may enable biologically plausible learning in the hidden recurrent layer during persistent task perturbation, we implement a previously proposed, temporally and spatially local learning rule, Random Feedback Local Online (RFLO) learning (Murray, 2019).

$$\tau \dot{B}_{ji} = \left( -B_{ji} + \Phi'(h_j)a_i^{prev} \right) \tag{3}$$

$$\dot{W}_{ji}^h = \eta_2 W_{jk}^{fb}\epsilon_k B_{ji} \tag{4}$$

where $\eta_2$ is the adaptation learning rate and $a_i^{prev}$ is hidden activity from the previous timestep i.e. $(t-1)$. Thus, the change in weights is proportional to the eligibility trace $B_{ji}$ and the feedback signal received $W_{jk}^{fb}\epsilon_k$. Note that when the same weight matrix $\boldsymbol{W^{fb}}$ is used for both control and learning, we denote this by adding $+c$ to the respective learning rule. For further biological realism, the learning batch size is fixed to 1 and the weight updates are applied online i.e. after every presented example.

## 3 Results

### 3.1 Recurrent neural networks with feedback control adapt well to task perturbation using feedback control and local learning rules

First, we pre-train the networks with feedback control on the *instructed delay center-out-reaching* task, where the networks need to produce 2D sigmoidal velocity profiles based on the target and the *go* signal input given (see Section 2 for details). Once trained, we test their ability to adapt to task perturbations using the visuo-motor rotation (VR) paradigm. Here, we rotate the target positions by 30° around the centre axis, creating a mismatch between the learned relationship between the input and the target. As previously shown in Feulner et al. (2022), the networks with feedback control show online adaptation to such a perturbation during ongoing movement in the first trial (Figure 1a). With persistent perturbation, the networks with feedback control show further improvement in the task performance, where the adapted network readout is independent of feedback control (Figure 1b).

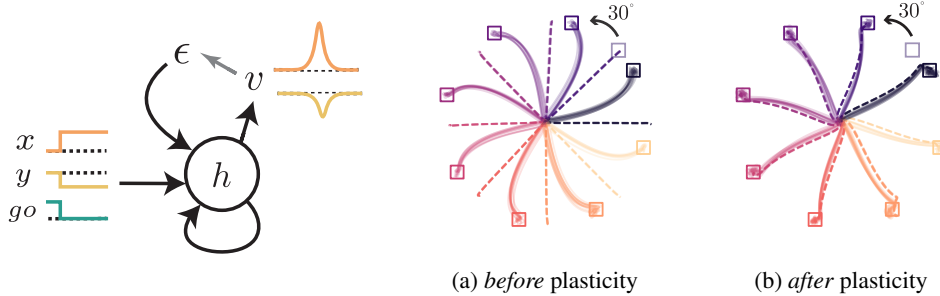

(a) *before* plasticity      (b) *after* plasticity

Figure 1: **Recurrent neural networks with feedback control feature rapid adaptation to *acute* task perturbation and can learn the *persistent* task perturbation using a local learning rule.** A recurrent neural network with feedback control is first pre-trained on produce a 2D sigmoidal velocity profile based on a target and *go* cue input input, and then tested on a 30° target rotation. (a) The pre-trained network adapts to the perturbation during the first trial with feedback control (full-line). The dashed line denotes the same network without feedback control. (b) The network further improves its performance during persistent perturbation with feedback-driven local learning (Feulner et al., 2022)

### 3.2 Feedback control approximates the true gradient w.r.t. networks activations during task perturbation

What happens during task perturbation in the networks with feedback control that allows for such rapid adaptation? In Figure 2a, we show in networks with feedback control, the feedback contribution to the overall network output increases during perturbation (see A.7 for full analysis details). Thus, during the task perturbation, the network activity is increasingly driven by the feedback signal. The average magnitude of this drive is proportional to the movement perturbation magnitude (Figure 2b), as the feedback drive is used to correct the recurrent network activity during perturbed trials. To further understand the mechanism behind this rapid movement correction, we investigate the alignment of the online feedback signal $W^{fb}\epsilon^{t-1}$ with the true first-order gradient w.r.t. the network

activations. We show that during predicted movement (between $t = 50$ and $t = 100$), the online feedback signal injected into the ongoing network dynamics approximates well the optimal, global trial gradient (Figure 2c). This adaptive behaviour is not typically observed in vanilla recurrent neural networks, where any performance feedback is *"locked"* i.e. not available to the network activity until the end of the whole trial, and even then only indirectly so through an update in the network weights. Thus, with feedback control, the feedback is *"unlocked"* and corrects the network activity in real-time.

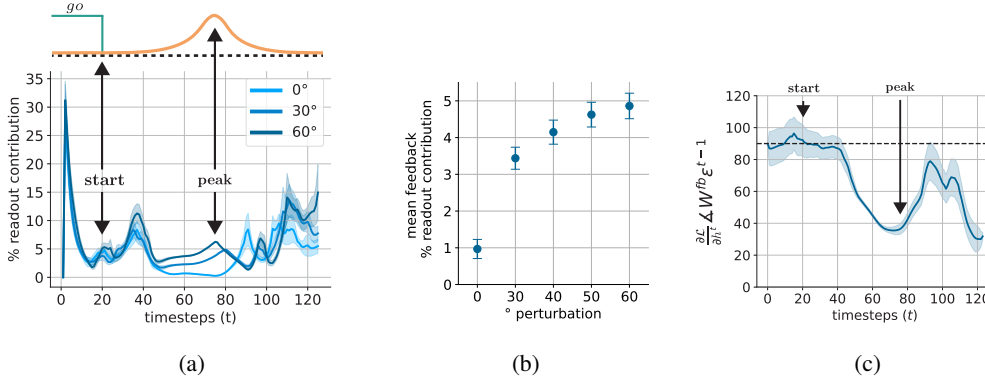

Figure 2: **Feedback control modulates rapid adaptation during task perturbation through approximate learning in the activity space.** (a) Increasing task perturbations degree ($0 \rightarrow 30 \rightarrow 60$) increases the relative feedback contribution to the network readout during a single task trial, which included a complete reaching movement. **start** denotes the start of the movement, and **peak** denotes the velocity peak. The shaded region denotes the standard error of the mean over 20 network seeds. (b) The average magnitude of the feedback drive is proportional to the movement perturbation magnitude. (c) The cosine similarity (reported in degrees) of the online feedback signal injected into the ongoing network dynamics and the optimal, global trial gradient w.r.t. activations during a single task trial. The shaded region denotes the standard error of the mean over 20 network seeds.

### 3.3 Feedback control *and* local, feedback-driven learning enable rapid network adaptation during persistent task perturbation

We next focus on network adaptation during persistent perturbation with feedback-driven plasticity *only* in the recurrent layer weights. Here, plasticity is governed by the temporally and spatially local learning rule that uses the learned feedback weights to project errors from the readout layer back to the recurrent layer. In addition to guiding credit assignment for the recurrent network weights, recurrent network activations are also concurrently influenced by feedback control. Thus, we refer to this as RFLO with control, i.e. RFLO+c (for more details, see Section 2.4). Such plasticity further improves the network performance during persistent adaptation (Figure 3a, blue line, Feulner et al. (2022)). Moreover, the network becomes independent of feedback control after a limited number of trials (Figure 3a, grey line). In Figure 3b, we show the final test performance without feedback control (i.e. control OFF) for the different learning algorithms at varying perturbation degree magnitudes. Using this simple learning rule with online feedback control during learning (RFLO+c), networks show performance similar to those trained with online BPTT on minor perturbations (Figure 3b). However, with larger perturbations, networks trained using local learning rules show an increasing performance gap with online BPTT (Figure 3b).

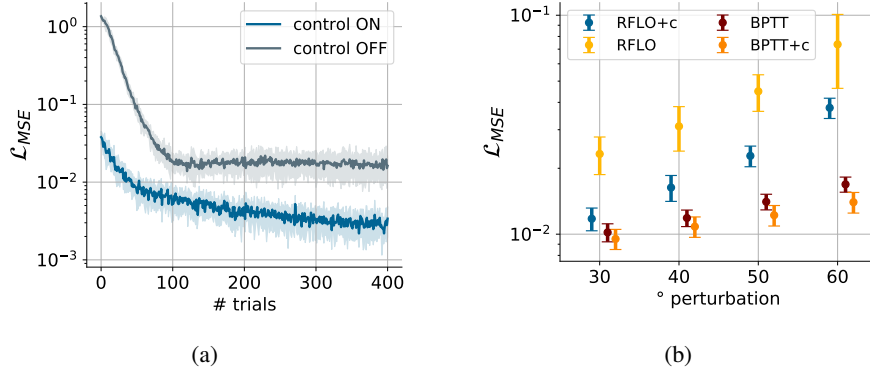

(a)                                                             (b)

Figure 3: **Feedback control aids local learning in the recurrent network layer.** (a) Train loss as a function of trial number during persistent perturbation with feedback-driven local learning. Grey line denotes the performance of the same network tested without feedback control, showing that the network relies less on feedback control with learning. The shaded regions denote the standard error of the mean over 10 random network seeds. (b) The final performance (after 1000 trials, without feedback control) of recurrent networks trained with RFLO with (+c, blue) or without (yellow) feedback control during learning with varying degrees of persistent perturbation. The performance of same networks trained with online BPTT with (+c, orange) or without (red) feedback. The error bars denote the standard error of the mean over 10 random network seeds.

## 3.4 Feedback control enables accurate recurrent weight adaptation during task perturbation

What makes online recurrent network adaptation so hard in the first place, and how might feedback control help to guide learning? To answer this question, we outline the computational requirements of the true online gradient w.r.t. the network activations and the feedback-driven local learning rule (Equation 4) using the notation from Marschall et al. (2020). Using online BPTT (Werbos, 1990), i.e. Real-Time Recurrent Learning (RTRL) (Williams & Zipser, 1989), one can decompose the total loss $\mathcal{L}_t$ w.r.t. parameters $\boldsymbol{W}$ using the chain rule as follows:

$$\sum_t \frac{\partial \mathcal{L}_t}{\partial \boldsymbol{W}} = \sum_t \frac{\partial \mathcal{L}_t}{\partial \boldsymbol{a}_t} \frac{\partial \boldsymbol{a}_t}{\partial \boldsymbol{W}} \tag{5}$$

where $\frac{\partial \mathcal{L}_t}{\partial \boldsymbol{a}_t} = \bar{\boldsymbol{c}}_t$ is the credit assignment vector and $\frac{\partial \boldsymbol{a}_t}{\partial \boldsymbol{W}} = \boldsymbol{M}_t$ is the influence matrix (Marschall et al., 2020). Using RTRL, the credit assignment vector $\bar{\boldsymbol{c}}$ is *immediate* as it is simply calculated by backpropagating the immediate error $\epsilon^t$ through the derivative of the output function $F_{\boldsymbol{W}}^o$ [1]. However, the influence matrix $\boldsymbol{M}$ requires the recursive computation of the past network states using the network Jacobian $\boldsymbol{J_t} = \partial \boldsymbol{a}_t / \partial \boldsymbol{a}_{t-1}$ (for details, see Marschall et al. (2020)).

In this work, similar to previous work (Feulner et al., 2022; Bellec et al., 2020; Murray, 2019), $\boldsymbol{M}$ is approximated with an *eligibility trace* $\boldsymbol{B}$ (see Equation 4). However, as shown in Marschall et al. (2020), such a severe approximation of the Jacobian can fail to capture the true temporal dependencies, instead biasing the learning trajectory towards capturing only short-term dependencies present in the neural activity at the point of learning. Depending on the task and the recurrent matrix structure, this can lead to the capturing of spurious correlations and a significant generalization gap (Liu et al., 2022). However, in networks with feedback control during task perturbation, the network activities are increasingly driven by the *present* feedback signal (Figure 2a), and thus less so by their *past*, recurrent state. We can quantify this further by calculating the norm of the Jacobian of the current network hidden activity w.r.t. past hidden activity with and without feedback control during task perturbation, which gives us an indication of current's networks sensitivity to its *past* activity vs the *present*, immediate feedback error. This is shown in Figure 4a, where we see that there is a drop in sensitivity to the past activity in the networks with feedback control during increasing task perturbation. Thus, as the network relies less on the past and more on the present, and is therefore "uncoupled", we expect the Jacobian approximation in the eligibility trace to the more accurate.

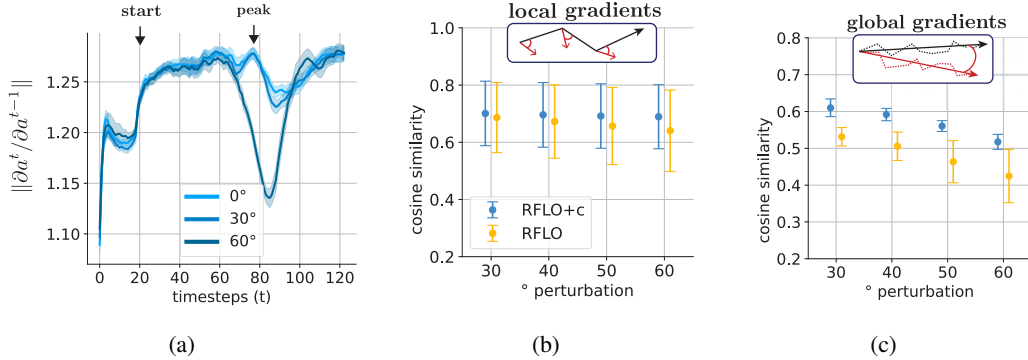

Figure 4: **Feedback increases the accuracy of local learning.** (a) The norm of the Jacobians of the network activities at time $t$ w.r.t. to activities at a previous timestep during a single trial. The shaded regions denote the standard deviation over 10 random network seeds. (b) The mean local alignment of the feedback-driven local learning rule with the true local gradient during task adaptation. The error bars denote the standard deviation over 10 seeds. The error bars denote the standard deviation over 10 random network seeds. (c) The mean global alignment of the feedback-driven local learning rule with the true global gradient during task adaptation. The error bars denote the standard deviation over 10 random network seeds.

We empirically validate this hypothesis by comparing the gradients of the local learning rule with that of online BPTT during task adaptation, with (RFLO+c) and without (RFLO) feedback control. Note that we calculate both the local, i.e. at each learning step, and global, i.e. after all the learning steps, gradient alignments. We show that while there is only a minor difference in the local gradient alignements, this compounds at the global level, where the local learning rule in the networks with feedback control aligns better with the true, global gradient than without feedback control (Figure 4b-c). This may explain the better performance of the networks with feedback control during task adaptation (Figure 3b), as the local learning rule is more accurate.

### 3.5 Feedback control enables efficient recurrent weight adaptation during task perturbation

To further understand the impact of feedback control on weight-based learning, we next focus on the relative magnitude of weight updates during task adaptation. Specifically, for each learning episode consisting of a limited number of trials, we can calculate the magnitude of final, global weight update $\|\Delta \boldsymbol{W}_{global}\|$, as well as the total combined magnitude of the online, local weight updates $\sum \|\Delta \boldsymbol{W}_{local}\|$. The ratio of the two quantities allows us to quantify the efficiency of the online weight updates during task adaptation. For instance, the less erratic the online weights updates are, the more efficient the learning trajectory, and our calculated ration will be closer to 1. Despite the relatively minor difference in local gradient accuracy (Figure 4b), we show that our pre-defined "efficiency ratio" is significantly different between the network with and without feedback control during learning, at all tested perturbation magnitudes (Figure 5a). Note that we don't observe this difference with more accurate gradients i.e. with BPTT (Figure 10b), but we do also observe it on an additional toy task (see Figure 11). Thus, with online feedback control, local weight updates are more efficient and less sensitive to the noise inherent to online learning using pseudo-gradients.

What could be the mechanism behind this marked increase in efficiency of the online weights updates with feedback control? In our networks, the feedback-controlled network activities are more constrained during task perturbation, where the network is initially more driven by the feedback signal that approximately aligns with their first-order gradient (see Figure 2c), irrespective of any weight based learning. This can be seen as a form of local adaptive inference in the activity space, or "prospective configuration" (Song et al., 2022), that can *guide* subsequent update steps in the weight space. Furthermore, the efficiency of learning is often linked to the network's sensitivity to its higher-order gradients, which equips it with better navigation of pathological curvatures in the objective function (Martens et al., 2010). To assess this in our networks, we calculate the alignment of the online feedback control signal injected into the network dynamics with the exact gradient used in the canonical $2^{nd}$ order optimization scheme, Newton's method. Here, the gradient is defined as $(\boldsymbol{H} - \gamma \boldsymbol{I})^{-1}\boldsymbol{g}$, where $\boldsymbol{H}$ is the Hessian w.r.t. the network activations, $\gamma$ is some small constant that re-conditions it and $\boldsymbol{g}$ is the $1^{st}$ order gradient. We show that during peaks of movement error, the feedback control signal injected into the ongoing network dynamics is also sensitive to their $2^{nd}$

order gradient (Figure5b, see also Figure 10a). Thus, feedback control also *implicitly* injects adaptive, second-order gradient information into the network dynamics, which may explain the magnitude of the increased efficiency observed in online weight updates during task adaptation.

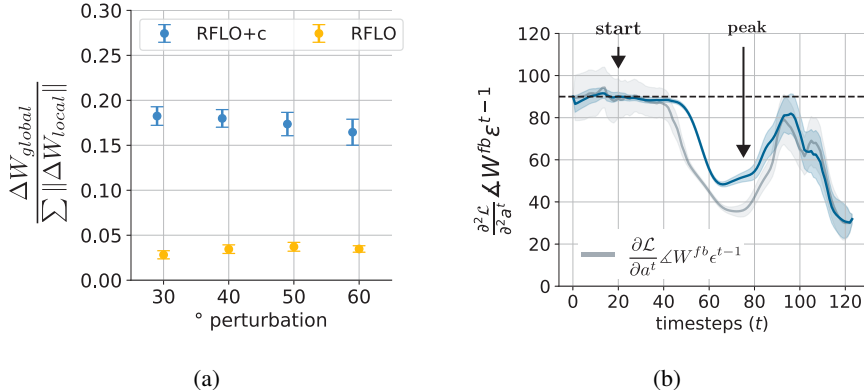

(a)                                                (b)

Figure 5: **Feedback increases the efficiency of local learning.** (a) The mean norm or efficiency ratio of the total, global weight update with that of the online, local weight update during task adaptation at various perturbation degrees. The error bars denote the standard deviation over 10 random network seeds. (b) The cosine similarity between the online feedback control signal injected into the ongoing network dynamics and the second-order gradient (calculated using the Newton method) w.r.t. activations (blue) during a single trial. The grey line denotes the first-order gradient as in Figure 2c. The shaded region denotes the standard error of the mean over 10 network seeds.

## 4   Discussion

**Summary.**   Although feedback connections are known to influence forward network activity in biological networks, the interaction of such feedback-driven activity for control with weight-based learning is less understood. In this work, we investigated the mechanistic properties of recurrent neural networks pre-trained with online feedback control using BPTT and tested on a task perturbation, which can also be seen as task modification, originally proposed in Feulner et al. (2022). We show that the online adaptation of such networks during task perturbations is due to (1) the increased feedback drive of the network and (2) the approximate alignment of the feedback signal with the true first and second-order gradient w.r.t. the network activations. Furthermore, we show that the consequence of such control is that the temporally and spatially local weight updates during task adaptation are both more accurate, robust and efficient.

**Feedback control and learning.**   Our results are in line with recent theoretical work showing that injecting control signals into network activity may aid local, biologically plausible learning in systems at equilibrium (Meulemans et al., 2022b). Here, we build on this work by showing that such strict equilibrium conditions can be relaxed in the presence of learned online feedback control during an ongoing network trial. Moreover, we show that the feedback control integration in the activity space also *implicitly* injects adaptive, second-order gradient information into the network dynamics, which may explain the magnitude of the increased efficiency observed in online weight updates during task adaptation. This is also in line with recent theoretical (Innocenti et al., 2023; Alonso et al., 2022) work linking network inference in predictive coding networks with the second-order, trust region optimization methods (Conn et al., 2000; Dauphin et al., 2014; Yuan, 2015). Here, unlike in standard line-search methods, a "safe" region in the loss landscape is first determined, before the actual weight updates within this region are calculated and applied. As in Innocenti et al. (2023), we show that in our networks this "safe" region could be implicitly calculated through adaptive, feedback-driven inference in the activity space. Importantly, in our networks this is done without an additional relaxation phase nor any explicit, second-order gradient calculation, and thus without the additional, often prohibitive, computational cost associated with such methods compared to a standard forward and backward network pass in BPTT.

**Limitations & future work.** In this work, we demonstrate the benefits of feedback control on a simple, biologically relevant task in single layer RNN. Further work is needed to investigate the generality of our findings to a wider range of tasks and network architectures. Moreover, we remain agnostic to the exact biological implementation of the feedback signal (probably via cerebellum, see Pemberton et al. (2022)) and the local learning rule (but see Aceituno et al. (2023)). Finally, the feedback form used in this work is a simple, linear projection of the signed error signal back to the network, and we anticipate that this may not hold in many brain regions. We believe it would be interesting both for computational neuroscience *and* machine learning to investigate the impact of less explicit feedback mechanism. Thus, extending our results to learning conditions with increased generality is an important direction for future work.

## 5   Impact statement

This paper presents work whose goal is to advance our understanding of fundamental principles of biological motor control. We recognize that this could have far-reaching implications in the future, particularly in the fields of robotics and prosthetics. Beyond these applications, this research may also enhance our understanding of motor disorders, potentially leading to improved rehabilitation strategies. Careful ethical considerations throughout our research are vital to proactively navigate the benefits and potential challenges these advancements may introduce to society.

## 6   Acknowledgements

This work was supported by the Wellcome Trust [grants 200790/Z/16/Z and 219995/Z/19/Z], the Simons Foundation [grant 564408], and EPSRC [grant EP/R035806/1]. We gratefully acknowledge the computational resources and support provided by the Imperial College Research Computing Service (http://doi.org/10.14469/hpc/2232). We also thank Dr. Tudor Berariu for the insightful discussions throughout this project and the reviewers for their constructive feedback, both of which greatly improved the quality of this paper.

## Footnotes

[1]In this work, $\bar{\boldsymbol{c}}_t$ is approximated with a pre-learned feedback matrix $\boldsymbol{W}^{fb}$, which through pre-training approximately aligns with the transpose of $\boldsymbol{W}^o$.

# References

Aceituno, P. V., Farinha, M. T., Loidl, R., and Grewe, B. F. Learning cortical hierarchies with temporal hebbian updates. *Frontiers in Computational Neuroscience*, 17:1136010, 2023.

Alemi, A., Machens, C., Deneve, S., and Slotine, J.-J. Learning nonlinear dynamics in efficient, balanced spiking networks using local plasticity rules. In *Proceedings of the AAAI conference on artificial intelligence*, volume 32, 2018.

Alonso, N., Millidge, B., Krichmar, J., and Neftci, E. O. A theoretical framework for inference learning. *Advances in Neural Information Processing Systems*, 35:37335–37348, 2022.

Bellec, G., Scherr, F., Subramoney, A., Hajek, E., Salaj, D., Legenstein, R., and Maass, W. A solution to the learning dilemma for recurrent networks of spiking neurons. *Nature communications*, 11(1): 3625, 2020.

Bengio, Y. How auto-encoders could provide credit assignment in deep networks via target propagation. *arXiv preprint arXiv:1407.7906*, 2014.

Bourdoukan, R. and Deneve, S. Enforcing balance allows local supervised learning in spiking recurrent networks. *Advances in Neural Information Processing Systems*, 28, 2015.

Briggs, F. and Usrey, W. M. Corticogeniculate feedback and visual processing in the primate. *The Journal of physiology*, 589(1):33–40, 2011.

Conn, A. R., Gould, N. I., and Toint, P. L. *Trust region methods*. SIAM, 2000.

Dauphin, Y. N., Pascanu, R., Gulcehre, C., Cho, K., Ganguli, S., and Bengio, Y. Identifying and attacking the saddle point problem in high-dimensional non-convex optimization. *Advances in neural information processing systems*, 27, 2014.

De Pasquale, R. and Sherman, S. M. Synaptic properties of corticocortical connections between the primary and secondary visual cortical areas in the mouse. *Journal of Neuroscience*, 31(46): 16494–16506, 2011.

Denève, S., Alemi, A., and Bourdoukan, R. The brain as an efficient and robust adaptive learner. *Neuron*, 94(5):969–977, 2017.

Douglas, R. J. and Martin, K. A. Neuronal circuits of the neocortex. *Annu. Rev. Neurosci.*, 27: 419–451, 2004.

Felleman, D. J. and Van Essen, D. C. Distributed hierarchical processing in the primate cerebral cortex. *Cerebral cortex (New York, NY: 1991)*, 1(1):1–47, 1991.

Feulner, B., Perich, M. G., Miller, L. E., Clopath, C., and Gallego, J. A. Feedback-based motor control can guide plasticity and drive rapid learning. *bioRxiv*, pp. 2022–10, 2022.

Fyall, A. M., El-Shamayleh, Y., Choi, H., Shea-Brown, E., and Pasupathy, A. Dynamic representation of partially occluded objects in primate prefrontal and visual cortex. *Elife*, 6:e25784, 2017.

Gauss, C. F. *Theoria motus corporum coelestium in sectionibus conicis solem ambientium*, volume 7. FA Perthes, 1877.

Gilbert, C. D. and Li, W. Top-down influences on visual processing. *Nature Reviews Neuroscience*, 14(5):350–363, 2013.

Gilra, A. and Gerstner, W. Predicting non-linear dynamics by stable local learning in a recurrent spiking neural network. *Elife*, 6:e28295, 2017.

Girard, P., Hupé, J., and Bullier, J. Feedforward and feedback connections between areas v1 and v2 of the monkey have similar rapid conduction velocities. *Journal of neurophysiology*, 85(3): 1328–1331, 2001.

He, K., Zhang, X., Ren, S., and Sun, J. Delving deep into rectifiers: Surpassing human-level performance on imagenet classification. In *Proceedings of the IEEE international conference on computer vision*, pp. 1026–1034, 2015.

Hochreiter, S. and Schmidhuber, J. Long short-term memory. *Neural computation*, 9(8):1735–1780, 1997.

Innocenti, F., Singh, R., and Buckley, C. Understanding predictive coding as a second-order trust-region method. In *ICML Workshop on Localized Learning (LLW)*, 2023.

Jordan, R. and Keller, G. B. Opposing influence of top-down and bottom-up input on excitatory layer 2/3 neurons in mouse primary visual cortex. *Neuron*, 108(6):1194–1206, 2020.

Keller, A. J., Roth, M. M., and Scanziani, M. Feedback generates a second receptive field in neurons of the visual cortex. *Nature*, 582(7813):545–549, 2020.

Kingma, D. P. and Ba, J. Adam: A method for stochastic optimization. *arXiv preprint arXiv:1412.6980*, 2014.

Krakauer, J. W., Pine, Z. M., Ghilardi, M.-F., and Ghez, C. Learning of visuomotor transformations for vectorial planning of reaching trajectories. *Journal of neuroscience*, 20(23):8916–8924, 2000.

Lee, D.-H., Zhang, S., Fischer, A., and Bengio, Y. Difference target propagation. In *Machine Learning and Knowledge Discovery in Databases: European Conference, ECML PKDD 2015, Porto, Portugal, September 7-11, 2015, Proceedings, Part I 15*, pp. 498–515. Springer, 2015.

Lillicrap, T. P. and Santoro, A. Backpropagation through time and the brain. *Current opinion in neurobiology*, 55:82–89, 2019.

Lillicrap, T. P., Santoro, A., Marris, L., Akerman, C. J., and Hinton, G. Backpropagation and the brain. *Nature Reviews Neuroscience*, 21(6):335–346, 2020.

Liu, Y. H., Smith, S., Mihalas, S., Shea-Brown, E., and Sümbül, U. Cell-type–specific neuromodulation guides synaptic credit assignment in a spiking neural network. *Proceedings of the National Academy of Sciences*, 118(51):e2111821118, 2021.

Liu, Y. H., Ghosh, A., Richards, B., Shea-Brown, E., and Lajoie, G. Beyond accuracy: generalization properties of bio-plausible temporal credit assignment rules. *Advances in Neural Information Processing Systems*, 35:23077–23097, 2022.

Manita, S., Suzuki, T., Homma, C., Matsumoto, T., Odagawa, M., Yamada, K., Ota, K., Matsubara, C., Inutsuka, A., Sato, M., et al. A top-down cortical circuit for accurate sensory perception. *Neuron*, 86(5):1304–1316, 2015.

Marschall, O., Cho, K., and Savin, C. A unified framework of online learning algorithms for training recurrent neural networks. *The Journal of Machine Learning Research*, 21(1):5320–5353, 2020.

Martens, J. et al. Deep learning via hessian-free optimization. In *Icml*, volume 27, pp. 735–742, 2010.

Meulemans, A., Carzaniga, F., Suykens, J., Sacramento, J., and Grewe, B. F. A theoretical framework for target propagation. *Advances in Neural Information Processing Systems*, 33:20024–20036, 2020.

Meulemans, A., Tristany Farinha, M., García Ordóñez, J., Vilimelis Aceituno, P., Sacramento, J., and Grewe, B. F. Credit assignment in neural networks through deep feedback control. *Advances in Neural Information Processing Systems*, 34:4674–4687, 2021.

Meulemans, A., Farinha, M. T., Cervera, M. R., Sacramento, J., and Grewe, B. F. Minimizing control for credit assignment with strong feedback. In *International Conference on Machine Learning*, pp. 15458–15483. PMLR, 2022a.

Meulemans, A., Zucchet, N., Kobayashi, S., Von Oswald, J., and Sacramento, J. The least-control principle for local learning at equilibrium. *Advances in Neural Information Processing Systems*, 35:33603–33617, 2022b.

Mozer, M., Chauvin, Y., and Rumelhart, D. Backpropagation: Theory, architectures, and applications. *Chapter A Focused Backpropagation Algorithm for Temporal Pattern Recognition, L. Erlbaum Associates Inc., Hillsdale, NJ, USA*, pp. 137–169, 1995.

Murray, J. M. Local online learning in recurrent networks with random feedback. *Elife*, 8:e43299, 2019.

Pascanu, R., Mikolov, T., and Bengio, Y. On the difficulty of training recurrent neural networks. In *International conference on machine learning*, pp. 1310–1318. Pmlr, 2013.

Payeur, A., Guerguiev, J., Zenke, F., Richards, B. A., and Naud, R. Burst-dependent synaptic plasticity can coordinate learning in hierarchical circuits. *Nature neuroscience*, 24(7):1010–1019, 2021.

Pemberton, J., Chadderton, P., and Costa, R. P. Cerebellar-driven cortical dynamics enable task acquisition, switching and consolidation. *bioRxiv*, pp. 2022–11, 2022.

Perich, M. G., Gallego, J. A., and Miller, L. E. A neural population mechanism for rapid learning. *Neuron*, 100(4):964–976, 2018.

Peters, A. and Payne, B. R. Numerical relationships between geniculocortical afferents and pyramidal cell modules in cat primary visual cortex. *Cerebral cortex*, 3(1):69–78, 1993.

Peters, A., Payne, B. R., and Budd, J. A numerical analysis of the geniculocortical input to striate cortex in the monkey. *Cerebral Cortex*, 4(3):215–229, 1994.

Podlaski, B. and Machens, C. K. Biological credit assignment through dynamic inversion of feedforward networks. *Advances in Neural Information Processing Systems*, 33:10065–10076, 2020.

Przybyszewski, A. W. Vision: Does top-down processing help us to see? *Current Biology*, 8(4): R135–R139, 1998.

Robinson, A. J. and Fallside, F. *The utility driven dynamic error propagation network*, volume 11. University of Cambridge Department of Engineering Cambridge, 1987.

Rumelhart, D. E., Hinton, G. E., and Williams, R. J. Learning representations by back-propagating errors. *nature*, 323(6088):533–536, 1986.

Sacramento, J., Ponte Costa, R., Bengio, Y., and Senn, W. Dendritic cortical microcircuits approximate the backpropagation algorithm. *Advances in neural information processing systems*, 31, 2018.

Scellier, B. and Bengio, Y. Equilibrium propagation: Bridging the gap between energy-based models and backpropagation. *Frontiers in computational neuroscience*, 11:24, 2017.

Song, Y., Millidge, B., Salvatori, T., Lukasiewicz, T., Xu, Z., and Bogacz, R. Inferring neural activity before plasticity: A foundation for learning beyond backpropagation. *bioRxiv*, pp. 2022–05, 2022.

Thoroughman, K. A. and Shadmehr, R. Learning of action through adaptive combination of motor primitives. *Nature*, 407(6805):742–747, 2000.

Werbos, P. J. Backpropagation through time: what it does and how to do it. *Proceedings of the IEEE*, 78(10):1550–1560, 1990.

Whittington, J. C. and Bogacz, R. An approximation of the error backpropagation algorithm in a predictive coding network with local hebbian synaptic plasticity. *Neural computation*, 29(5): 1229–1262, 2017.

Williams, R. J. and Zipser, D. A learning algorithm for continually running fully recurrent neural networks. *Neural computation*, 1(2):270–280, 1989.

Yuan, Y.-x. Recent advances in trust region algorithms. *Mathematical Programming*, 151:249–281, 2015.

# A Detailed methods

## A.1 Task

To create a synthetic *instructed delay centre-out reaching* task as experimentally performed in monkeys (Perich et al., 2018), we set both the trajectory start ($p^{start} \in \mathbb{R}^2$) and end point ($p^{end} \in \mathbb{R}^2$) by randomly sampling each coordinate independently from a uniform distribution $U(-6, 6)$ and interpolate the corresponding velocity trajectories $[v_x, v_y]$ with a sigmoid function parametrized by $\kappa = 10/s$ (Equation 6).

$$f(t) = \frac{1}{1 + exp(-t\kappa)} \tag{6}$$

This creates natural, bell-shaped reaching velocity profiles. Each trial lasts $\approx 1.25$ s (125 steps) and features a (fixed) instructed delay period $t^{go}$ of 0.2 s, included in the network as a Heaviside step function that is non-zero during the delay period. Thus, the input and output task dimensionalities are 3 and 2 respectively. The radius of the circle for the *center-out-reach* sub-task used in visualisations is set to 5 cm.

## A.2 Network

Here, we use recurrent neural networks with 400 hidden units (but also see Figure 9). The weights ($\boldsymbol{W}^{in}, \boldsymbol{W}^h, \boldsymbol{W}^o, \boldsymbol{W}^{fb}$) and biases ($\boldsymbol{b}^h, \boldsymbol{b}^o$) of the networks are initialized from a uniform distribution $U(-1/\sqrt{l}, 1/\sqrt{l})$ (He et al., 2015) where $l$ is either the number of layer parameters (for $\boldsymbol{W}^{in}, \boldsymbol{W}^h, \boldsymbol{W}^{fb}, \boldsymbol{b}^h$) or the dimensionality of the output (for $\boldsymbol{W}^o, \boldsymbol{b}^o$). The hidden network activity $x$ is integrated using Equation 1, with $dt = 10$ ms and $\tau = 50$ ms.

## A.3 Pre-training

We pre-train the network parameters using the Adam optimizer (Kingma & Ba, 2014) with learning rate $\eta_1 = 0.001(\beta_1 = 0.9, \beta_2 = 0.999)$, batch size of 256 and L2 regularization for network parameters ($\beta = 1e^{-3}$) and activations ($\gamma = 2e^{-3}$) for 5000 independently drawn data batches i.e. trials. The total norm of the gradients is clipped to 0.2. Here, we eliminate the separate parameter training phases used in Feulner et al. (2022) and keep all network parameters plastic throughout the initial training. We also perturb the network output during the entire pre-training stage by adding brief (0.1 s) velocity pulses with an amplitude of 10 cm/s at random in 25% of the trials.

## A.4 Fine-tuning

During persistent perturbation, we use the SGD optimizer and the batch size of 1, and modify the weight updates after each trial timestep as per Equation 4. As per the learning rate sweep shown in Figure 8, the fine-tuning learning rates $\eta_2$ used for each learning algorithm are: $1e^{-4}$ (BPTT+c), $5e^{-6}$ (BPTT) and $5e^{-5}$ (RFLO, RFLO+c). Note that for the analysis in the Figure 4 we use $\eta_2 = 1e^{-5}$.

## A.5 Computational requirements

We ran all our experiments using NVIDIA GeForce GPUs (RTX 2080 Ti). The code, adapted from (Feulner et al., 2022), is available at: https://github.com/klarakaleb/feedback-control.

## A.6 Key differences from the modelling setup in Feulner et al. (2022)

- **Task Simplification:** We simplify the task by fixing the *go* signal timing, facilitating a clearer analysis of within-trial adaptation mechanisms.
- **Curriculum Streamlining:** We use a single-phase training curriculum instead of the three-phase approach, again focusing on the core mechanisms of interest.
- **Pre-training batch size increase:** We use a larger batch size than in Feulner et al. (2022) (20 vs 246), for faster convergence of networks with and without feedback.

- **Online feedback:** We provide truly online feedback, removing the biologically-motivated feedback delay used in Feulner et al. (2022).
- **Shorter Individual Trials:** We shorten the individual trial lengths (300 to 125) to improve the distribution of target values - in Feulner et al. (2022), most of the target values are set to 0 due to the steepness of the velocity sigmoid.
- **Local Learning Rule:** We improve the biological plausibility of the learning rule by making it fully online (instead of accumulated) with a continuous eligibility trace (see Equation 4). We also add $\Phi'(a_j)$ to the eligibility trace, which makes it equivalent to that of RFLO learning (Murray, 2019) and increases its stability with larger perturbations (see Figure 6 and 7).

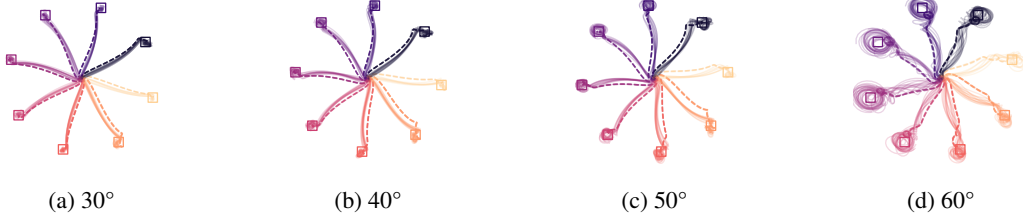

|  (a) 30° | (b) 40° | (c) 50° | (d) 60° |

Figure 6: Performance of networks trained using the local rule from Feulner et al. (2022) with increasing degrees of perturbation.

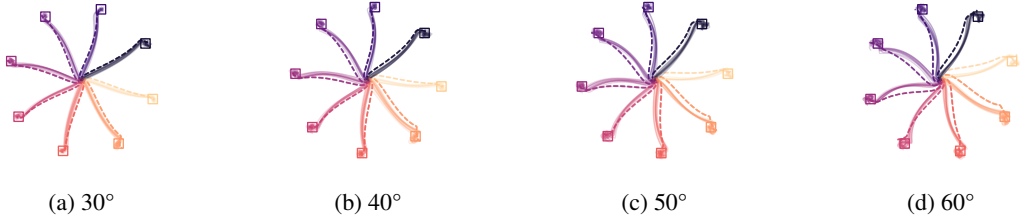

|  (a) 30° | (b) 40° | (c) 50° | (d) 60° |

Figure 7: Performance of networks trained using RFLO learning from Murray (2019) with increasing degrees of perturbation.

## A.7 Analysis details

To quantify the percentage readout contribution of the immediate feedback control signal in Figure 2a, we revisit Equations 1 and 2 that govern the network hidden and readout dynamics, respectively. Combining the two into a single equation, we get:

$$v_k = \sum_j W_{kj}^o a_j + b_k^o \tag{7}$$

$$= \sum_j W_{kj}^o \Phi(\dot{h_j}) + b_k^o \tag{8}$$

$$= \sum_j W_{kj}^o \Phi \frac{dt}{\tau} \left( -h_j + \sum_i W_{ji}^{in} s_i + \sum_i W_{ji}^h \Phi(h_i) + \sum_i W_{ji}^{fb} \epsilon_i + b_j^h \right) + b_k^o \tag{9}$$

Then, the percentage of the immediate feedback control signal % contribution to the network output is calculated as:

$$\text{percentage feedback contribution} = \sum_j [h_j > 0] W_{kj}^o \left( \frac{dt}{\tau} \sum_i W_{ji}^{fb} \epsilon_i \right) \times \frac{100}{\sum_j W_{kj}^o a_j} \tag{10}$$

Note that for Figure 2a we report an average value over $k$.

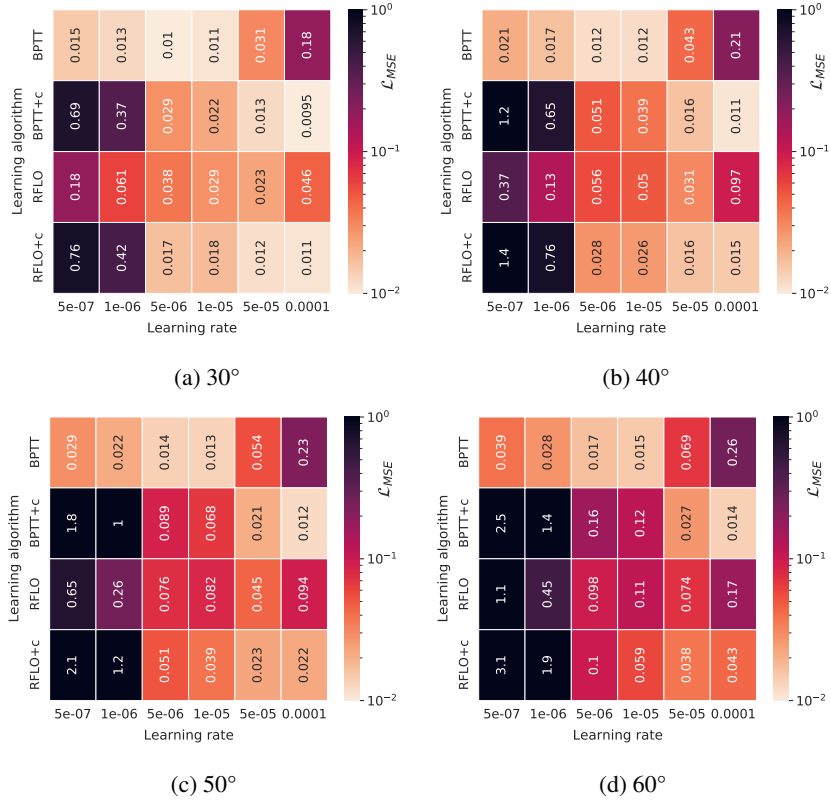

Figure 8: Adaptation learning rate sweep using different learning algorithms with increasing degrees of persistent perturbation.

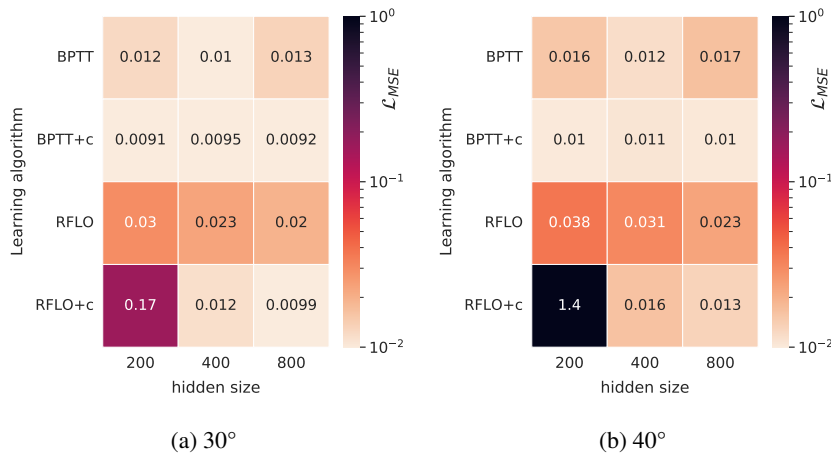

Figure 9: Hidden network size sweep for the different learning algorithms with increasing degrees of persistent perturbation. The adaptation learning rates for each learning algorithm are: $1e^{-4}$ (BPTT+c), $5e^{-6}$ (BPTT) and $5e^{-5}$ (RFLO, RFLO+c), as per Figure 8.

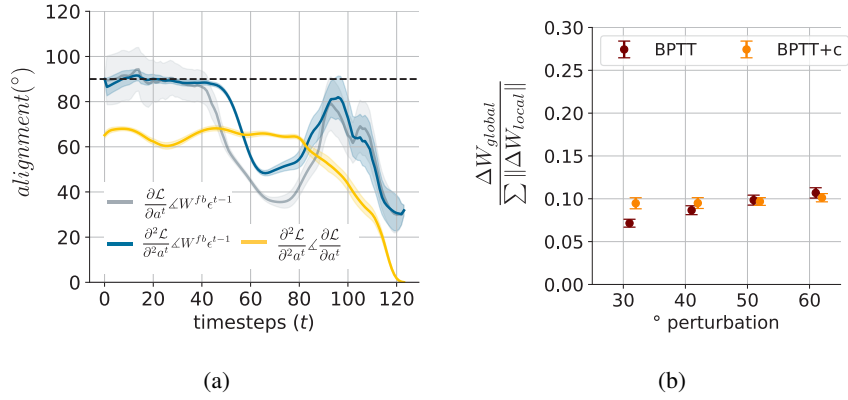

(a)  (b)

Figure 10: (a) Baseline measured alignment between the 1st and 2nd order gradient (yellow line), as in Figure 2c and 5b. (b) Learning efficiency for BPTT with (orange) and without (red) control, as in Figure 5a.

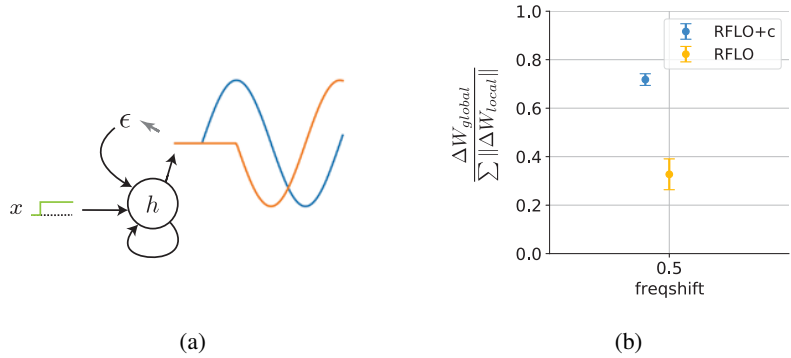

(a)  (b)

Figure 11: (a) *Sine and Cosine wave task*: given some input frequency $f$ sampled uniformly from $U(1,4)$, the network has to generate a 2D output containing both the Sine and the Cosine with the same given frequency. Note we mask the first quarter of the period for the Cosine wave to ensure task smoothness. The corresponding perturbation in this task is a frequency shift - here, the desired generated frequency is increased by some constant. (b) Learning efficiency during persistent perturbation (+0.5 Hz) for the *sine and cosine wave task* for RFLO with (blue) and without (yellow) control, as in Figure 5a. Here, $\eta_2$ is set to $1e^{-5}$.

